# A mixture model for the evolution of gene expression in non-homogeneous datasets

**Gerald Quon[1], Yee Whye Teh[2], Esther Chan[3], Timothy Hughes[3], Michael Brudno[1,3], Quaid Morris[3]**
[1]Department of Computer Science, and [3]Banting and Best Department of Medical Research, University of Toronto, Canada,
[2]Gatsby Computational Neuroscience Unit, University College London, United Kingdom
{gerald.quon,quaid.morris}@utoronto.ca

## Abstract

We address the challenge of assessing conservation of gene expression in complex, non-homogeneous datasets. Recent studies have demonstrated the success of probabilistic models in studying the evolution of gene expression in simple eukaryotic organisms such as yeast, for which measurements are typically scalar and independent. Models capable of studying expression evolution in much more complex organisms such as vertebrates are particularly important given the medical and scientific interest in species such as human and mouse. We present Brownian Factor Phylogenetic Analysis, a statistical model that makes a number of significant extensions to previous models to enable characterization of changes in expression among highly complex organisms. We demonstrate the efficacy of our method on a microarray dataset profiling diverse tissues from multiple vertebrate species. We anticipate that the model will be invaluable in the study of gene expression patterns in other diverse organisms as well, such as worms and insects.

## 1 Introduction

High-throughput functional data is emerging as an indispensible resource for generating a complete picture of genome-wide gene and protein function. Currently, gene function is often inferred through sequence comparisons with genes of known function in other species, though sequence similarity is no guarantee of shared biological function. Gene duplication, one of the primary forces of genomic evolution, often gives rise to genes with high sequence similarity but distinct biological roles [1]. Differences in temporal and spatial gene expression patterns have also been posited to explain phenotypic differences among animals despite a surprisingly large degree of gene sequence similarity [2]. This observation and the increasingly wide availability of genome-wide gene expression profiles from related organisms has motivated us to develop statistical models to study the evolution of gene expression along phylogenies, in order to identify lineages where gene expression and therefore gene function is likely to be conserved or diverged.

Comparing gene expression patterns between distantly related multi-cellular organisms is challenging because it is difficult to collect a wide range of functionally matching tissue samples. In some cases, matching samples simply may not exist because some organismal functions have been redistributed among otherwise homologous organs. For example, processes such as B-cell development are performed by both distinct and overlapping sets of tissues: primarily bone marrow in mammals; Bursa of Fabricus and bone marrow in birds; and likely kidney, spleen, and/or thymus in teleost fish (who lack bone marrow) [3]. Matching samples can also be hard to collect because anatomical arrangements of some of the queried organisms make isolation of specific tissues virtually impossible. For example, in frog, the kidneys are immediately adjacent to the ovaries and are typically covered in oocytes. By allowing tissue samples to be mixed and heterogeneous, though functionally related, it

becomes possible to compare expression patterns describing a much larger range of functions across a much larger range of organisms.

Current detailed statistical models of expression data assume measurements from matched samples in each organism. As such, comparative studies of gene expression to date have either resorted to simple, non-phylogenetic measures to compare expression patterns [4], or restricted their comparisons to single-cellular organisms [5] or clearly homologous tissues in mammals [6].

Here, we present Brownian Factor Phylogenetic Analysis (BFPA), a new model of gene expression evolution that removes the earlier limitations of matched samples, therefore allowing detailed comparisons of expression patterns from the widely diverged multi-cellular organisms. Our model takes as input expression profiles of orthologous genes in multiple present-day organisms and a phylogenetic tree connecting those organisms, and simultaneously reconstructs the expression profiles for the ancestral nodes in the phylogenetic tree while detecting links in the phylogeny where rapid change of the expression profile has occurred.

We model the expression data from related organisms using a mixture of Gaussians model related to a mixture of constrained factor analyzers [7]. In our model, each mixture component represents a different pattern of conservation and divergence of gene expression along each link of the phylogenetic tree. We assume a constrained linear mapping between the heterogeneous samples in different organisms and fit this mapping using maximum likelihood. We show that by expanding the amount of expression data that can be compared between species, our model generates more useful information for predicting gene function and is also better able to reconstruct the evolutionary history of gene expression as evidenced by its increased accuracy in reconstructing gene expression levels.

## 2   Previous work

Recent evolutionary models of gene expression treat it as a quantitative (i.e. real-valued) trait and model evolutionary change in expression levels as a Brownian motion process [8, 9]. Assuming Brownian motion, a given gene's expression level $x_s$ in a child species $s$ after a divergence time $t_s$ from an ancestral species $\pi(s)$ is predicted to be Gaussian distributed with a mean $x_{\pi(s)}$ equal to the gene's expression level in the ancestor and variance $\sigma^2 t_s$:

$$x_s \sim N(x_{\pi(s)}, \sigma^2 t_s) \tag{1}$$

where $\sigma^2$ represents the expected rate of change per unit time. The ancestor-child relationships are specified using a phylogeny, such as that shown in Figure 1a for the vertebrates. The leaves of the phylogeny are associated with present-day species and the internal branch points with shared ancestors. The exact position of the root of the phylogeny (not shown in the figure, but somewhere along branch "T") cannot be established without additional information, and the outgroup species "T" is often used in place of the root of the tree. Nonetheless, the rooted phylogeny can be interpreted as a directed Gaussian graphical model, e.g. Figure 1b, whose nodes are variables representing expression levels in the corresponding species and whose directed edges point from immediate ancestors to their children species. The conditional probability distribution (CPD) at each node is given by Equation 1.

Typical uses of these evolutionary models are to compare different hypotheses about divergence times [8] or the structure of the phylogeny [9] by calculating the likelihood of the present-day expression levels under various hypotheses. To avoid assigning this prior over the root node and thus introducing bias [10], Felsenstein developed a method called restricted maximum likelihood (REML) [11], which specifies a distribution over the observed differences between present-day expression levels rather than the expression levels themselves.

## 3   Brownian Factor Phylogenetic Analysis: A model of expression evolution

In the following section, we propose changes to the Brownian motion model that not only allow for unmatched tissue samples, but also leverage the change observed in expression levels across multiple genes in order to classify genes into different patterns of expression evolution. We use $x_s^i$ to indicate the hidden expression profile of the $i$-th gene (out of $N$ ortholog groups) in species $s$.

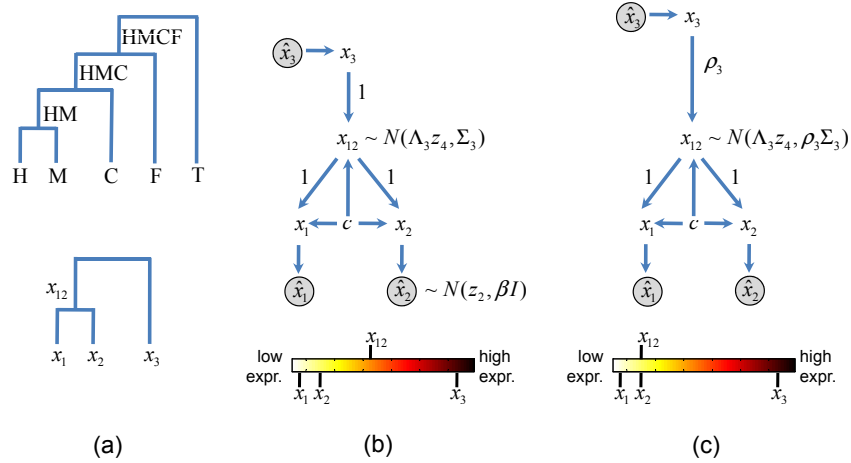

(a)                                          (b)                                          (c)

Figure 1: *Our statistical model and associated species phylogenies.* (a) The phylogeny of the species measured in our dataset of human (H), mouse (M), chicken (C), frog (F), and tetraodon (T), as well as an example phylogeny of three hypothetical species $x_1$, $x_2$, and $x_3$ used to illustrate our model. (b) Our statistical model showing how the outgroup species $x_3$ and its corresponding observed expression levels $\hat{x}_3$ is used as a gene expression prior. Edge weights on the graph depict scaling factors applied to the variance terms $\Sigma$, which are specified by each conservation pattern $c$. 1 denotes no scaling on that branch, whereas $\rho > 1$ depicts a longer, and thus unconserved, branch. This particular conservation pattern represents a phylogeny where all species have conserved expression. The scale on the bottom shows hypothetical values for $x_1$, $x_2$, and $x_3$, as well as the inferred value for $x_{12}$. (c) The same model except applied to a conservation pattern where species $x_3$ is determined to exhibit significantly different expression levels (rapid change).

The input to our model are vectors of tissue-specific expression levels $\{\hat{x}_s^i\}_{i=1}^N$ for $N$ genes over present-day species $s \in \{P \cup o\}$; we distinguish the chosen outgroup species $o$ from the rest of the present-day species $P$. $\hat{x}_s^i \in \mathbb{R}^{d_s}$, where $d_s$ is the number of tissues in species $s$. The goal of our model is to infer each gene's corresponding pattern of gene expression evolution (conservation pattern) $\{c^i\}_{i=1}^N$ and latent expression levels $\{x_s^i\}_{i=1}^N$ for all species $s \in \{P \cup o \cup A\}$, where $A$ represents the internal ancestral species in the phylogenetic tree (Figure 1). The likelihood function $\mathcal{L} = P\left(\{\hat{x}_P^i, x_{P\cup o\cup A}^i, c^i\}_{i=1}^N | \{\hat{x}_o^i\}_{i=1}^N, \theta\right)$ is shown below, where $\pi(s)$ refers to the parent species of $s$, $\theta = (\Lambda, \Sigma, \beta, \rho, \gamma)$ are the model parameters, and $N(x; \mu, \Sigma)$ is the density of $x$ under a multivariate normal distribution with mean $\mu$ and covariance $\Sigma$:

$$\mathcal{L} = \prod_i \left[ \left( \prod_{s \in P \cup A} P(x_s^i | x_{\pi(s)}^i, c^i, \theta) \right) \times \left( \prod_{s \in P} P(\hat{x}_s^i | x_s^i, \beta) \right) \right] P(x_o^i | \hat{x}_o^i, \beta) P(c^i | \gamma)$$

$$P(x_s^i | x_{\pi(s)}^i, c^i = K_j, \theta) = N(x_s^i; \Lambda_s x_{\pi(s)}^i, \rho_s^{K_{j,s}} \Sigma_s) \tag{2}$$

$$P(\hat{x}_s^i | x_s^i, \beta) = N(\hat{x}_s^i; x_s^i, \beta_s) \tag{3}$$

$$P(c^i = K_j | \gamma) = \gamma_j \tag{4}$$

*Modeling branch lengths.* Equation 2 reflects the central assumption of Brownian motion models [8, 9, 10] described in Equation 1, extended in two ways. BFPA extends this concept in two directions. First, we constrain all variances $\Sigma_s$ to be diagonal in order to estimate tissue-specific drift rates, as tissues are known to vary widely in expression divergence rates [12]. Secondly, we note that in studying a diverse lineage such as vertebrates, we expect to see large changes in expression for genes that have diverged in function, as compared to genes of conserved function. We therefore model the drift of a gene's expression levels along each branch of the tree as following one of two rates: a slow rate, reflecting a functional constraint, and a fast rate, reflecting neutral or selected change. Correspondingly, for each branch of the phylogenetic tree above the species $s$, we define two rate parameters, $\rho_s^2$ or $\rho_s^1$, termed a short and long branch respectively ($\rho_s^2 < \rho_s^1$). We fix $\rho_s^2 = 1.0$ and initialize $\rho_s^1$ to a much larger value to maintain this relationship during learning, thus modeling fast-moving genes as outliers. Our method of modelling constrained and unconstrained change as scalar multiples of a common variance is similar to the discrete gamma method [13].

*Linear relationship between ancestral and child tissues.* We model tissues of child species as linear combinations of ancestral tissues. The matrix of coefficients $\Lambda_s$ that relate expression levels in the child species' tissues to that of its parent species is heavily constrained to leverage our prior understanding of the relationships of specific tissues [14]. To construct $\Lambda_s$, pairs of tissues that were clearly homologous (i.e. the heart) had their corresponding entry in $\Lambda_s$ fixed at 1, and all other entries in the same row set to zero. For the remaining tissues, literature searches were conducted to determine which groups of tissues had broadly related function (i.e. immune tissues), and those entries were allowed to vary from zero. All other entries were constrained to be zero.

*Distinguishing intra- and inter-species variation.* Equation 3 relates the observed expression levels of present-day species to the noiseless, inferred expression levels of the corresponding hidden nodes of each observed species. The variance factor $\beta_s$ is an estimate of the variation expected due to noise in the array measurements, and are estimated via maximum likelihood using multiple identical probes present on each microarray.

*Conservation pattern estimation.* Our goal is to identify different types of expression evolution, including punctuated evolution, fully conserved expression, or rapid change along all branches of the phylogeny. We model the problem as a mixture model of *conservation patterns*, in which each conservation pattern specifies either constrained or fast change along each branch of the tree. Each conservation pattern $K_j \in \{1,2\}^{|P \cup A|}$ specifies a configuration of $\rho_s^1$ or $\rho_s^2$ for each species $s$ ($K_{j,s} \in \{1,2\}$ specifies $\rho_s^{K_{j,s}}$). However, not all $2^{|P \cup A|}$ possible patterns of short and long branches can be uniquely considered. In particular, a tree containing at least one ancestor incident to two long branches and one short are ambiguous because this tree cannot be distinguished from the same tree with that ancestor incident to three long branches. As a post-processing step, we consider short branches in those cases to be long, and sum over such ambiguous trees, leaving a total of $J$ possible conservation patterns. Each pattern $K_j$ is assigned a prior probability $P(K_j) = \gamma_j$ that is learned, as reflected in Equation 4.

# 4   Inference

Because our graphical model contains no cycles, we can apply belief propagation to perform exact inference and obtain the posterior distributions $P(c^i = K_j | \hat{x}^i, \theta), \forall i, j$:

$$\delta_{ij} = P(c^i = K_j | \hat{x}^i, \theta) \propto \int P(x^i_{P \cup o \cup A}, \hat{x}^i_P, c^i = K_j | \hat{x}^i_o, \theta) \partial x^i_{P \cup o \cup A} \tag{5}$$

We can also estimate the distributions over expression levels of a species $s'$ as

$$P(x^i_{s'} | \hat{x}^i, \theta) \propto \sum_j \int P(x^i_{P \cup o \cup A}, \hat{x}^i_P, c^i = K_j | \hat{x}^i_o, \theta) \partial x^i_{P \cup o \cup A \setminus s'} \tag{6}$$

# 5   Learning

Applying the expectation maximization (EM) algorithm yields the following maximum likelihood estimates of the model parameters, where $\mathbb{E}_{s,s|K_j} = \mathbb{E}[x^i_s x^{i^T}_s | \hat{x}^i, c^i = K_j]$, $\mathbb{E}_{s,\pi(s)|K_j} = \mathbb{E}[x^i_s x^{i^T}_{\pi(s)} | \hat{x}^i, c^i = K_j]$, and $\mathbb{E}_{\pi(s),\pi(s)|K_j} = \mathbb{E}[x^i_{\pi(s)} x^{i^T}_{\pi(s)} | \hat{x}^i, c^i = K_j]$:

$$\hat{\Lambda}_s = \left( \sum_{i=1}^N \sum_{j=1}^J \frac{\delta_{ij}}{\rho_s^{K_{j,s}}} \mathbb{E}_{s,\pi(s)|K_j} \right) \left( \sum_{i=1}^N \sum_{j=1}^J \frac{\delta_{ij}}{\rho_s^{K_{j,s}}} \mathbb{E}_{\pi(s),\pi(s)|K_j} \right)^{-1} \tag{7}$$

$$\hat{\Sigma}_s = \frac{1}{N} \text{diag} \left\{ \sum_{i=1}^N \sum_{j=1}^J \frac{\delta_{ij}}{\rho_s^{K_{j,s}}} \left( \mathbb{E}_{s,s|K_j} - 2\Lambda_s \mathbb{E}_{s,\pi(s)|K_j}^T + \Lambda_s \mathbb{E}_{\pi(s),\pi(s)|K_j} \Lambda_s^T \right) \right\}$$

$$\hat{\rho}_s^k = \left( \sum_i \sum_j [K_{j,s} = k] \delta_{ij} \text{dim}(x^i_s) \right)^{-1} \left( \sum_i \sum_j [K_{j,s} = k] \delta_{ij} \times \right.$$
$$\left. \left( \text{tr}[\mathbb{E}_{s,s|K_j} \Sigma_s^{-1}] + \text{tr} \left[ \Lambda_s^T \Sigma_s^{-1} (-2\mathbb{E}_{s,\pi(s)|K_j} + \Lambda_s \mathbb{E}_{\pi(s),\pi(s)|K_j}) \right] \right) \right)$$

$$\hat{\gamma}_j = \frac{\sum_{i=1}^{N} \delta_{ij}}{N} \tag{8}$$

Although we have rooted the phylogeny using a present-day species rather than place a hypothetical root as has been done in previous Brownian motion models, these two models are related because they are equivalent under the condition that all samples are matched. First, note that in traditional Brownian motion models, the location of the root is arbitrary if one assumes a constant, improper prior over the root expression levels, since any choice of root would give rise to the same probability distribution over the expression levels. By using a present-day species with observed expression levels as the root node, we avoid integrating over this improper prior. Because the root node prior is constant, the likelihood of the other present-day species conditioned on this present-day root expression level is a constant times the likelihood of all present-day species expression levels. Our conditional model therefore assigns identical likelihoods and marginals as REML.

## 6    Results

We present the results of applying our model to a novel dataset consisting of gene expression measurements of 4770 genes with unique, unambiguous orthology, i.e., each of the 4770 genes is present in only a single copy, across the following five present-day organisms: human, mouse, chicken, frog, and tetraodon. The phylogeny related these species is shown in Figure 1 with nodes labelled by the first letter of the species name. We set Tetradon as the root, so $o = T$ and $P = \{H, M, C, F\}$ and we label the internal ancestors by concatenating the labels of their present-day descendants, so $A = \{HM, HMC, HMCF\}$.

Replicate microarray probe intensity measurements were taken for the 4770 genes across a total of 161 tissues (i.e., 322 microarrays in total) in the five organisms: 46 tissues from human, 55 from mouse, and 20 from each of the other three organisms. We applied a standard pre-processing pipeline to the array set to remove experimental artifacts and to transform the probe intensity measurements on each array to a common, variance-stabilized scale. Each array was first spatially detrended as described in [15]. Within a species, all arrays share the same probe set, so we applied VSN [16] to the arrays from each species to estimate an array-specific affine transform to transform the probe intensities to species-specific units. We next applied an arcsinh transform to the probe intensities to make the variance of the noise independent of the intensity measurement. For the final two pre-processing steps, we placed the transformed intensity measurements into a matrix for each species. The rows of this matrix correspond to genes and the columns are the measured tissues. First, to remove probe bias in the transformed intensities, we subtracted the row median from each element and then to attempt to transform measurements from different species to a common scale, we subtracted the column means from each element and divided by the column length.

First, we investigate the stability of our conservation pattern estimates by using parameters trained on different random subsamples of our genes. We then evaluate the predictive value of our algorithm BFPA using two tasks: a) predicting gene expression profiles in a new species given expression profiles in other species, and b) predicting Gene Ontology annotation using the conservation pattern inferred by our model.

To perform the stability experiments, we first randomly split the dataset into five subsets, and used each subset individually to train the model using 100 iterations of EM. We then estimated $P(c^i | \hat{x}_s^i, \theta)$ for the four other subsets of genes, and classified each gene into its most likely conservation pattern. Hence, each gene is classified four times by non-overlapping training sets. Figure 2 shows that the classifications are quite stable and that most genes are classified into few conservation patterns. Most genes that were uniquely classified into a single conservation pattern either were classified as fully (all) conserved or completely unconserved, resulting in relatively few high-confidence lineage-specific genes.

### 6.1    Functional associations of co-transcriptionally evolving genes

Pairs of genes exhibiting correlated expression also tend to perform similar function. This guilt-by-association principle is often used to initially assign putative functions to genes. For example, a popular method for analyzing gene expression datasets is to cluster genes based on the pairwise

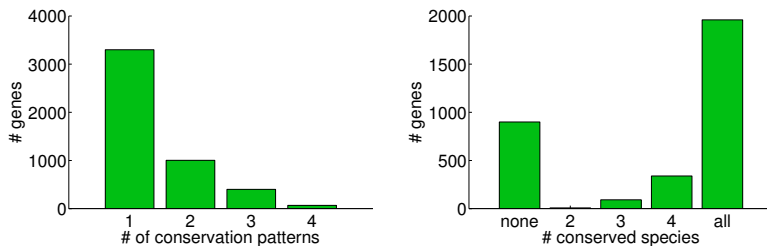

Figure 2: *Stability of conservation pattern assignments to genes.* (left) Each gene was placed into one of four bins, denoting the number of unique patterns it was classified into. Most genes were consistently classified into one conservation pattern for all four of its independent classifications. (right) For all genes uniquely classified into a single conservation pattern, the number of present-day species adjacent to conserved links was computed. Most genes were either classified as fully (all) conserved or completely unconserved.

Pearson correlation coefficient (PCC), then measure the enrichment of these clusters in Gene Ontology (GO) function and process annotations [17]. In this section, we introduce the evolutionary correlation coefficient (ECC), a simple modification of PCC to integrate model predictions, and examine whether genes with the same annotated function are more similar in rank according the ECC or PCC measures. ECC scales the positively-transformed PCC by the marginal probability of the genes following the same expression evolution, assuming independent evolution.

$$ECC(\hat{x}^i, \hat{x}^k) \;\;=\;\; \left(1 + PCC(\hat{x}^i, \hat{x}^k)\right) \sum_j P(c^i = j|\hat{x}^i, \theta) P(c^k = j|\hat{x}^k, \theta)$$

ECC can be applied using the output of either BFPA or the Brownian model. For the Brownian model, we trained and made predictions using only those matched samples in all five species. Those ten samples are the central nervous system (CNS), intestine, heart, kidney, liver, eye, muscle, spleen, stomach, and testis. We also introduce ECC-sequence, designed to measure the value of evolutionary information derived from sequence. First, the protein sequences of each gene were aligned using default parameters of MUSCLE [18]. These alignments were then inputted into PAML [19] together with the species tree shown in Figure 1 to estimate branch lengths. The PCC measure for each pair of genes was then scaled by the Pearson correlation coefficient of the branch lengths estimated by PAML to produce ECC-sequence.

For all models, we first used the ECC/PCC similarity metric for each gene to rank all other genes in order of expression similarity. We then apply the Wilcoxon Rank Sum test to evaluate whether genes with the same GO annotations, as annotated for the mouse ortholog, are significantly higher in rank than all other genes. For this analysis, we only considered GO Process categories which have at least one of the 4770 genes annotated in that category. We also removed all genes which were not annotated in any category, resulting in a total of 3319 genes and 4246 categories.

Figure 3 illustrates the distribution of smallest $p$-values achieved by each gene over all of their annotated functions. PCC is used as a baseline performance measure as it does not consider evolutionary information. We see that all evolutionary-based models outperform PCC in ranking genes with similar function much closer on average. ECC-sequence performs worse than PCC, suggesting that expression-based evolutionary metrics may provide additional information compared to those based on sequence. The relative performance of BFPA versus Brownian reflects an overall significant performance gap between our models and the existing ones. A control measure ECC-random is shown, which is computed by randomizing the gene labels of the data in each of the five organisms before learning. Finally, Brown+prior measures the performance of the Brownian model when the conservation pattern priors are allowed to be estimated, and performs better than the Brownian model but worse than BFPA, as expected. All differences between the distributions are statistically significant, as all pairwise $p$-values computed by the Kolmogorov-Smirnov test are less than $10^{-6}$.

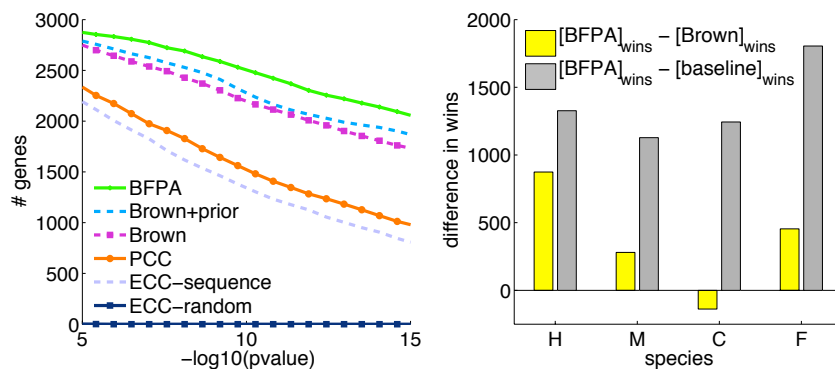

Figure 3: *Model performance.* (left) A reverse cumulative distribution plot of $p$-values obtained from applying the Wilcoxon Rank Sum test using either a PCC or ECC-based similarity metric. The smallest $p$-value achieved for each gene across all its annotated functions is used in the distribution. Position $(x, y)$ indicates that for $y$ genes, their $p$-value was less than $10^{-x}$. Higher lines on the graph translate into stronger associations between expression levels and gene function, which we interpret as better performance. (right) This graph shows the difference in the total number of expression values for which a particular method achieves the lowest error, sorted by species.

## 6.2 Reconstruction of gene expression levels

Here we report the performance of our model in predicting the expression level of a gene in each of human, mouse, chicken, and frog, given its expression levels in the other species. Tetraodon is not predicted because it acts as an outgroup in our model. The model was trained using 100 EM iterations on half of the dataset, which was then used to predict the expression levels for each gene in each species in the other half of the dataset, and vice versa. To create a baseline performance measure, we computed the error when using an average of the four other species to predict the expression level of a gene in the fifth species. We only compute predictions for the ten matched samples across all species so that we can compare errors made by our model against those of Brownian and the baseline, which require matched samples. Figure 3 shows that with the exception of the comparison against Brownian in chicken, BFPA achieves lower error than both Brownian and baseline in predicting expression measurements.

## 7 Discussion

We have presented a new model for the simultaneous evolution of gene expression levels across multiple tissues and organs. Given expression data from present-day species, our model can be used to simultaneously infer the ancestral expression levels of orthologous genes as well as determine where in the phylogeny the gene expression levels underwent substantial change. BFPA extends previous Brownian models [8, 9] by introducing a constrained factor analysis framework to account for complex tissue relationships between different species and by adapting the discrete gamma method [13] to model quantitative gene expression data. Our model performs better than other Brownian models in functional association and expression prediction experiments, demonstrating that the evolutionary history we infer better recovers the function of the gene. We have shown that this is in large part due to our ability to consider species-specific tissue measurements, a feature not implemented in any existing model to the best of our knowledge. We also showed that gene expression-based phylogenetic data may provide information not contained in sequence-based phylogenetic data in terms of helping predict the functional association of genes.

Our model has a number of other applications outside of using it to study the evolutionary history of gene expression. Our ability to identify genes with conserved expression across multiple species will help in the inference of gene function from annotated to non-annotated species because unconserved expression patterns indicate a likely change in the biological function of a gene. We also expect that by identifying species that share a conserved expression pattern, our model will aid in the

identification of transcriptional *cis*-regulatory elements by focusing the search for *cis*-elements to those species identified as conserved in expression.

While we have taken different profiled samples as representing different tissues, our methodology can be easily expanded to study evolutionary change in gene expression in response to different growth conditions or environmental stresses, as with those studied in [5]. Our methodology is also easily extendible to other model organisms for which there are genomes and expression data for multiple closely related species (e.g. yeast, worm, fly, plants). We anticipate that the results obtained will be invaluable in the study of genome evolution and identification of *cis*-regulatory elements, whose phylogeny should reflect that of the gene expression patterns.

All data used in this publication can be obtained by a request to the authors.

## References

[1] Li, W., Yang, J., Gu, X. (2005) Expression divergence between duplicate genes. *Trends Genet.*, **21**, 602-607.

[2] Chen, K., Rajewsky, N. (2007) The evolution of gene regulation by transcription factors and microRNAs. *Nature Rev. Genet.*, **8**, 93-103.

[3] Yergeau, D.A. *et al.* (2005) *bloodthirsty*, an RBCC/TRIM gene required for erythropoiesis in zebrafish. *Dev. Biol.,* **283**, 97-112.

[4] Stuart, J.M., Segal, E., Koller, D., Kim, S.K. (2003) A gene-coexpression network for global discovery of conserved genetic modules. *Science*, **302**, 249-255.

[5] Tirosh, I., Weinberger, A., Carmi, M., Barkai, N. (2006) A genetic signature of interspecies variations in gene expression. *Nat. Genet.*, **38**, 830-834.

[6] Khaitovich, P. *et al.* (2005) A neutral model of transcriptome evolution. *PLoS. Biol.*, **2**, 682-689.

[7] Ghahramani, Z., & Hinton, G.E. (1996) The EM algorithm for mixtures of factor analyzers. *Technical Report CRG-TR-96-2*, University of Toronto.

[8] Gu, X. (2004) Statistical framework for phylogenomic analysis of gene family expression profiles. *Genetics*, **167**, 531-542.

[9] Oakley, T.H. *et al.* (2005) Comparative methods for the analysis of gene-expression evolution: an example using yeast functional genomic data. *Mol. Biol. Evol.*, **22**, 40-50.

[10] Felsenstein, J. (2004) Inferring phylogenies. Sunderland (Massachusetts): Sinauer Associates. 664 p.

[11] Felsenstein, J. (1981) Evolutionary trees from gene-frequencies and quantitative characters - finding maximum likelihood estimates. *Evolution*, **35**, 1229-1242.

[12] Khaitovich *et al.* (2006) Evolution of primate gene expression. *Nat. Rev. Genet.*, **7**, 693-702.

[13] Yang, Z. (1994) Maximum likelihood phylogenetic estimation from DNA sequences with variable rates over sites: approximate methods. *J. Mol. Evol.*, **39**, 306-314.

[14] Kardong, K.V. (2006) Vertebrates: comparative anatomy, function, evolution. McGraw-Hill. 782 p.

[15] Zhang, W., Morris, Q.D. *et al.* (2004) The functional landscape of mouse gene expression. *J. Biol.*, **3**, 21.

[16] Huber, W. *et al.* (2002) Variance stabilization applied to microarray data calibration and to the quantification of differential expression. *Bioinformatics*, **18**, S96-104.

[17] The Gene Ontology Consortium. (2000) Gene Ontology: tool for the unification of biology. *Nature Genet.*, **25**, 25-29.

[18] Edgar, R.C. (2004) MUSCLE: multiple sequence alignment with high accuracy and high throughput. *Nucleic Acids Res.*, **32**, 1792-1797.

[19] Yang, Z. (2007) PAML 4: phylogenetic analysis by maximum likelihood. *Mol. Biol. Evol.*, **24**, 1586-1591.

